# Factorial Learning and the *EM* Algorithm

**Zoubin Ghahramani**
zoubin@psyche.mit.edu

Department of Brain & Cognitive Sciences
Massachusetts Institute of Technology
Cambridge, MA 02139

## Abstract

Many real world learning problems are best characterized by an interaction of multiple independent causes or factors. Discovering such causal structure from the data is the focus of this paper. Based on Zemel and Hinton's cooperative vector quantizer (CVQ) architecture, an unsupervised learning algorithm is derived from the Expectation–Maximization (EM) framework. Due to the combinatorial nature of the data generation process, the exact E-step is computationally intractable. Two alternative methods for computing the E-step are proposed: Gibbs sampling and mean-field approximation, and some promising empirical results are presented.

## 1 Introduction

Many unsupervised learning problems fall under the rubric of *factorial learning*—that is, the goal of the learning algorithm is to discover multiple independent causes, or factors, that can well characterize the observed data (Barlow, 1989; Redlich, 1993; Hinton and Zemel, 1994; Saund, 1995). Such learning problems often arise naturally in response to the actual process by which the data have been generated. For instance, images may be generated by combining multiple objects, or varying colors, locations, and poses, with different light sources. Similarly, speech signals may result from an interaction of factors such as the tongue position, lip aperture, glottal state, communication line, and background noises. The goal of factorial learning is to invert this data generation process, discovering a representation that will both parsimoniously describe the data and reflect its underlying causes.

A recent approach to factorial learning uses the Minimum Description Length (MDL) principle (Rissanen, 1989) to extract a compact representation of the input (Zemel, 1993; Hinton and Zemel, 1994). This has resulted in a learning architecture

called Cooperative Vector Quantization (CVQ), in which a set of vector quantizers cooperates to reproduce the input. Within each vector quantizer a competitive learning mechanism operates to select an appropriate vector code to describe the input. The CVQ is related to algorithms based on mixture models, such as soft competitive clustering, mixtures of experts (Jordan and Jacobs, 1994), and hidden Markov models (Baum et al., 1970), in that each vector quantizer in the CVQ is itself a mixture model. However, it generalizes this notion by allowing the mixture models to cooperate in describing features in the data set, thereby creating a distributed representations of the mixture components. The learning algorithm for the CVQ uses MDL to derive a cost function composed of a reconstruction cost (e.g. sum squared error), representation cost (negative entropy of the vector code), and model complexity (description length of the network weights), which is minimized by gradient descent.

In this paper we first formulate the factorial learning problem in the framework of statistical physics (section 2). Through this formalism, we derive a novel learning algorithm for the CVQ based on the Expectation–Maximization (EM) algorithm (Dempster et al., 1977) (section 3). The exact EM algorithm is intractable for this and related factorial learning problems—however, a tractable mean-field approximation can be derived. Empirical results on Gibbs sampling and the mean-field approximation are presented in section 4.

## 2   Statistical Physics Formulation

The CVQ architecture, shown in Figure 1, is composed of hidden and observable units, where the observable units, $\mathbf{y}$, are real-valued, and the hidden units are discrete and organized into vectors $\mathbf{s}_i$, $i = 1, \ldots, d$. The network models a data generation process which is assumed to proceed in two stages. First, a factor is independently sampled from each hidden unit vector, $\mathbf{s}_i$, according to its prior probability density, $\boldsymbol{\pi}_i$. Within each vector the factors are mutually exclusive, i.e. if $s_{ij} = 1$ for some $j$, then $s_{ik} = 0$, $\forall k \neq j$. The observable is then generated from a Gaussian distribution with mean $\sum_{i=1}^{d} W_i \mathbf{s}_i$.

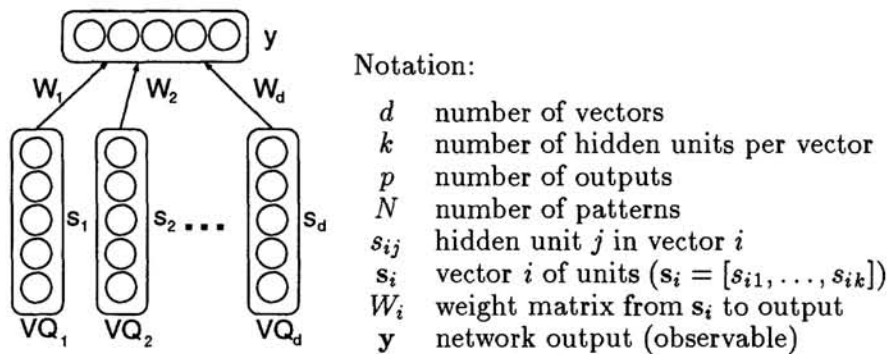

Notation:

| | |
|---|---|
| $d$ | number of vectors |
| $k$ | number of hidden units per vector |
| $p$ | number of outputs |
| $N$ | number of patterns |
| $s_{ij}$ | hidden unit $j$ in vector $i$ |
| $\mathbf{s}_i$ | vector $i$ of units ($\mathbf{s}_i = [s_{i1}, \ldots, s_{ik}]$) |
| $W_i$ | weight matrix from $\mathbf{s}_i$ to output |
| $\mathbf{y}$ | network output (observable) |

Figure 1. The factorial learning architecture.

Defining the energy of a particular configuration of hidden states and outputs as

$$\mathcal{H}(\mathbf{s}, \mathbf{y}) = \frac{1}{2}\|\mathbf{y} - \sum_{i=1}^{d} W_i \mathbf{s}_i\|^2 - \sum_{i=1}^{d}\sum_{j=1}^{k} s_{ij} \log \pi_{ij}, \tag{1}$$

the Boltzmann distribution

$$p(\mathbf{s}, \mathbf{y}) = \frac{1}{Z_{free}} \exp\{-\mathcal{H}(\mathbf{s}, \mathbf{y})\}, \qquad (2)$$

exactly recovers the probability model for the CVQ. The causes or factors are represented in the multinomial variables $\mathbf{s}_i$ and the observable in the multivariate Gaussian $\mathbf{y}$. The unclamped partition function, $Z_{free}$, can be evaluated by summing and integrating over all the possible configurations of the system to obtain

$$Z_{free} = \sum_s \int_y \exp\{-\mathcal{H}(\mathbf{s}, \mathbf{y})\} dy = (2\pi)^{p/2}, \qquad (3)$$

which is constant, independent of the weights. This constant partition function results in desirable properties, such as the lack of a Boltzmann machine-like sleep phase (Neal, 1992), which we will exploit in the learning algorithm.

The system described by equation (1)[1] can be thought of as a special form of the Boltzmann machine (Ackley et al., 1985). Expanding out the quadratic term we see that there are pairwise interaction terms between every unit. The evaluation of the partition function (3) tells us that when $\mathbf{y}$ is unclamped the quadratic term can be integrated out and therefore all $\mathbf{s}_i$ are independent. However, when $\mathbf{y}$ is clamped all the $\mathbf{s}_i$ become dependent.

## 3  The EM Algorithm

Given a set of observable vectors, the goal of the unsupervised learning algorithm is to find weight matrices such that the network is most likely to have generated the data. If the hidden causes for each observable where known, then the weight matrices could be easily estimated. However, the hidden causes cannot be inferred unless these weight matrices are known. This chicken-and-egg problem can be solved by iterating between computing the expectation of the hidden causes given the current weights and maximizing the likelihood of the weights given these expected causes—the two steps forming the basis of the Expectation–Maximization (EM) algorithm (Dempster et al., 1977).

Formally, from (2) we obtain the expected log likelihood of the parameters $\phi'$:

$$Q(\phi, \phi') = \langle -\mathcal{H}(\mathbf{s}, \mathbf{y}) - \log Z_{free} \rangle_{c,\phi} \qquad (4)$$

where $\phi$ denotes the current parameters, $\phi = \{W_i\}_{i=1}^d$, and $\langle \cdot \rangle_{c,\phi}$ denotes expectation given $\phi$ and the clamped observables. The E-step of EM consists of computing this expected log likelihood. As the only random variables are the hidden causes, this simplifies to computing the $\langle \mathbf{s}_i \rangle_c$ and $\langle \mathbf{s}_i \mathbf{s}_j^T \rangle_c$ terms appearing in the quadratic expansion of $\mathcal{H}$. Once these terms have been computed, the M-step consists of maximizing $Q$ with respect to the parameters. Setting the derivatives to zero we obtain a linear system,

$$\frac{\partial Q}{\partial W_j} = \mathbf{y} \langle \mathbf{s}_j \rangle_c^T - \sum_{i=1}^d W_i \langle \mathbf{s}_i \mathbf{s}_j^T \rangle_c = 0,$$

which can be solved via the normal equations,

$$\widehat{W}_{(d \times k \times p)} = \left[ \sum_N \langle \mathbf{ss}^T \rangle \langle \mathbf{ss}^T \rangle \right]^{-1}_{(d \times k)^2} \left[ \sum_N \langle \mathbf{ss}^T \rangle \langle \mathbf{s} \rangle \mathbf{y}^T \right]_{(d \times k \times p)}$$

where **s** is the vector of concatenated $\mathbf{s}_i$ and the subscripts denote matrix size.

For models in which the observable is a monotonic differentiable function of $\sum_i W_i \mathbf{s}_i$, i.e. generalized linear models, least squares estimates of the weights for the M-step can be obtained iteratively by the method of scoring (McCullagh and Nelder, 1989).

### 3.1   E-step: Exact

The difficulty arises in the E-step of the algorithm. The expectation of hidden unit $j$ in vector $i$ given pattern **y** is:

$$
\begin{aligned}
\langle s_{ij} \rangle_c &= P(s_{ij} = 1 | \mathbf{y}; W) \propto P(\mathbf{y} | s_{ij} = 1; W) \pi_{ij} \\
&\propto \sum_{j_1=1}^{k} \cdots \sum_{j_h \neq i=1}^{k} \cdots \sum_{j_d=1}^{k} P(\mathbf{y} | s_{ij} = 1, s_{1j_1} = 1, \ldots, s_{dj_d} = 1; W) \pi_{ij}
\end{aligned}
$$

To compute this expectation it is necessary to sum over all possible configurations of the other hidden units. If each vector quantizer has $k$ hidden units, each expectation has time complexity of $\mathcal{O}(k^{d-1})$, i.e. $\mathcal{O}(Nk^d)$ for a full E-step. The exponential time is due inherently to the cooperative nature of the model—the setting of one vector only determines the observable if all the other vectors are fixed.

### 3.2   E-step: Gibbs sampling

Rather than summing over all possible hidden unit patterns to compute the exact expectations, a natural approach is to approximate them through a Monte Carlo method. As with Boltzmann machines, the CVQ architecture lends itself well to Gibbs sampling (Geman and Geman, 1984). Starting from a clamped observable **y** and a random setting of the hidden units $\{\mathbf{s}_j\}$, the setting of each vector is updated in turn stochastically according to its conditional distribution $\mathbf{s}_i \sim p(\mathbf{s}_i | \mathbf{y}, \{\mathbf{s}_j\}_{j \neq i}; W)$. Each conditional distribution calculation requires $k$ forward passes through the network, one for each possible state of the vector being updated, and $k$ Gaussian distance calculations between the resulting predicted and clamped observables. If all the probabilities are bounded away from zero this process is guaranteed to converge to the equilibrium distribution of the hidden units given the observable. The first and second-order statistics, for $\langle \mathbf{s}_i \rangle_c$ and $\langle \mathbf{s}_i \mathbf{s}_j^T \rangle_c$ respectively, can be collected using the $s_{ij}$'s visited and $p(\mathbf{s}_i | \mathbf{y}, \{\mathbf{s}_j\}_{j \neq i}; W)$ calculated during this sampling process. These estimated expectations are then used in the E-step.

### 3.3   E-step: Mean-field approximation

Although Gibbs sampling is generally much more efficient than exact calculations, it too can be computationally demanding. A more promising approach is to approximate the intractable system with a tractable mean-field approximation (Parisi, 1988), and perform the E-step calculation on this approximation. We can write the

negative log likelihood minimized by the original system as a difference between the clamped and unclamped free energies:

$$
\begin{aligned}
\text{Cost} &= -\log p(\mathbf{y}|W) = -\log \sum_{\mathbf{s}} p(\mathbf{y}, \mathbf{s}|W) \\
&= -\log \sum_{\mathbf{s}} \exp\{-\mathcal{H}(\mathbf{y}, \mathbf{s})\} + \log \sum_{\mathbf{s}} \int_{\mathbf{y}} \exp\{-\mathcal{H}(\mathbf{y}, \mathbf{s})\} d\mathbf{y} \\
&= F_{cl} - F_{free}
\end{aligned}
$$

The mean-field approximation allows us to replace each free energy in this cost with an upper bound approximation $\text{Cost}^{MF} = F_{cl}^{MF} - F_{free}^{MF}$. Unfortunately, a difference of two upper bounds is not generally an upper bound, and therefore minimizing $\text{Cost}^{MF}$ in, for example, mean-field Boltzmann machines does not guarantee that we are minimizing an upper bound on Cost. However, for the factorial learning architectures described in this paper we have the property that $F_{free}$ is constant, and therefore the mean-field approximation of the cost is an upper bound on the exact cost.

The mean-field approximation can be obtained by approximating the probability density given by (1) and (2) by a completely factorized probability density:

$$
\tilde{p}(\mathbf{s}, \mathbf{y}) = \frac{1}{(2\pi)^{p/2}} \exp\{-\frac{1}{2}\|\mathbf{y} - \boldsymbol{\mu}\|^2\} \prod_{i,j} m_{ij}^{s_{ij}}
$$

In this approximation all units are independent: the observables are Gaussian distributed with mean $\boldsymbol{\mu}$ and each hidden unit is binomially distributed with mean $m_{ij}$. To obtain the mean-field approximation we solve for the mean values that minimize the Kullback-Leibler divergence $\mathcal{KL}(p, \tilde{p}) \equiv E_{\tilde{p}}[\log \tilde{p}] - E_{\tilde{p}}[\log p]$.

Noting that: $E_{\tilde{p}}[s_{ij}] = m_{ij}$, $E_{\tilde{p}}[s_{ij}^2] = m_{ij}$, $E_{\tilde{p}}[s_{ij}s_{kl}] = m_{ij}m_{kl}$, and $E_{\tilde{p}}[s_{ij}s_{ik}] = 0$, we obtain the mean-field fixed point equations

$$
\mathbf{m}_i = \text{softmax}\left(W_i^T(\mathbf{y} - \hat{\mathbf{y}}) + W_i^T W_i(\mathbf{m}_i - 1/2)\right), \tag{5}
$$

where $\hat{\mathbf{y}} \equiv \sum_i W_i \mathbf{m}_i$. The softmax function is the exponential normalized over the $k$ hidden units in each $\mathbf{m}_i$ vector. The first term inside the softmax has an intuitive interpretation as the projection of the error in the observable onto the weights of the hidden unit vector $i$. The more a hidden unit can reduce this error, the higher its mean. The second term arises from the fact that $E_{\tilde{p}}[s_{ij}^2] = m_{ij}$ and not $E_{\tilde{p}}[s_{ij}^2] = m_{ij}^2$. The means obtained by iterating equation (5) are used in the E-step by substituting $\mathbf{m}_i$ for $\langle \mathbf{s}_i \rangle_c$ and $\mathbf{m}_i \mathbf{m}_j^T$ for $\langle \mathbf{s}_i \mathbf{s}_j^T \rangle_c$.

## 4   Empirical Results

Two methods, Gibbs sampling and mean-field, have been provided for computing the E-step of the factorial learning algorithm. There is a key empirical question that needs to be answered to determine the efficiency and accuracy of each method. For Gibbs sampling it is important to know how many samples will provide robust estimates of the expectations required for the E-step. It is well known that for stochastic Boltzmann machines the number of samples needed to obtain good

estimates of the gradients is generally large and renders the learning algorithm prohibitively slow. Will this architecture suffer from the same problem? For mean-field it is important to know the loss incurred by approximating the true likelihood. We explore these questions by presenting empirical results on two small unsupervised learning problems.

The first benchmark problem consists of a data set of $4 \times 4$ greyscale images generated by a combination of two factors: one producing a single horizontal line and the other, a vertical line (Figure 2a; cf. Zemel, 1993). Using a network with 2 vectors of 4 hidden units each, both the Gibbs sampling and mean-field EM algorithms converge on a solution after about a dozen steps (Figure 2b). The solutions found resemble the generative model of the data (Figure 2c & d).

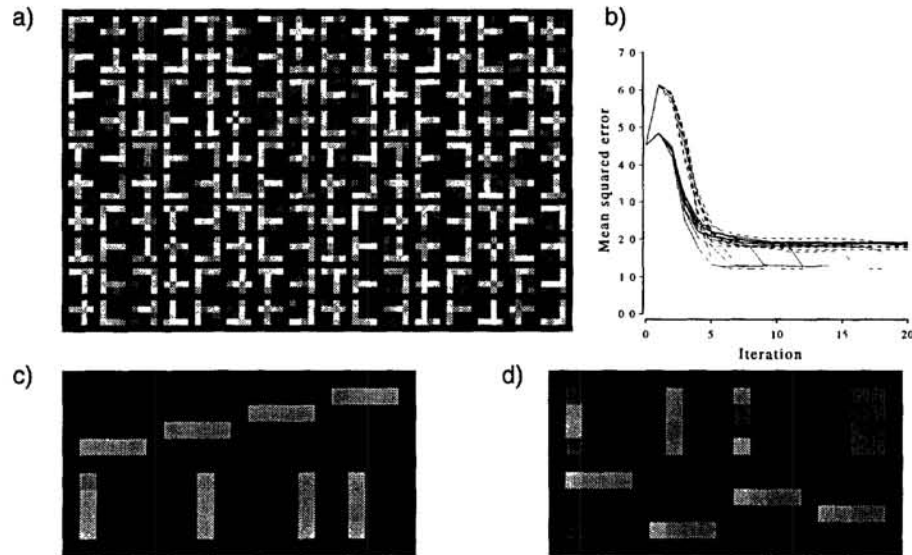

Figure 2. Lines Problem. a) Complete data set of 160 patterns. b) Learning curves for Gibbs (solid) and mean-field (dashed) forms of the algorithm. c) A sample output weight matrix after learning (MSE=1.20). The top vector of hidden units has come to represent horizontal lines, and the bottom, vertical lines. d) Another typical output weight matrix (MSE=1.78).

The second problem consists of a data set of $6 \times 6$ images generated by a combination of three shapes—a cross, a diagonal line, and an empty square—each of which can appear in one of 16 locations (Figure 3a). The data set of 300 out of 4096 possible images was presented to a network with the architecture shown in Figure 3b. After 30 steps of EM, each consisting of 5 Gibbs samples of each hidden unit, the network reconstructed a representation that approximated the three underlying causes of the data—dedicating one vector mostly to diagonal lines, one to hollow squares, and one to crosses (Figure 3c).

To assess how many Gibbs samples are required to obtain accurate estimates of the expectations for the E-step we repeated the lines problem varying the number of samples. Clearly, as the number of samples becomes large the Gibbs E-step becomes exact. Therefore we expect performance to asymptote at the performance of the exact E-step. The results indicate that, for this problem, 3 samples are sufficient to achieve ceiling performance (Figure 4). Surprisingly, a single iteration of the

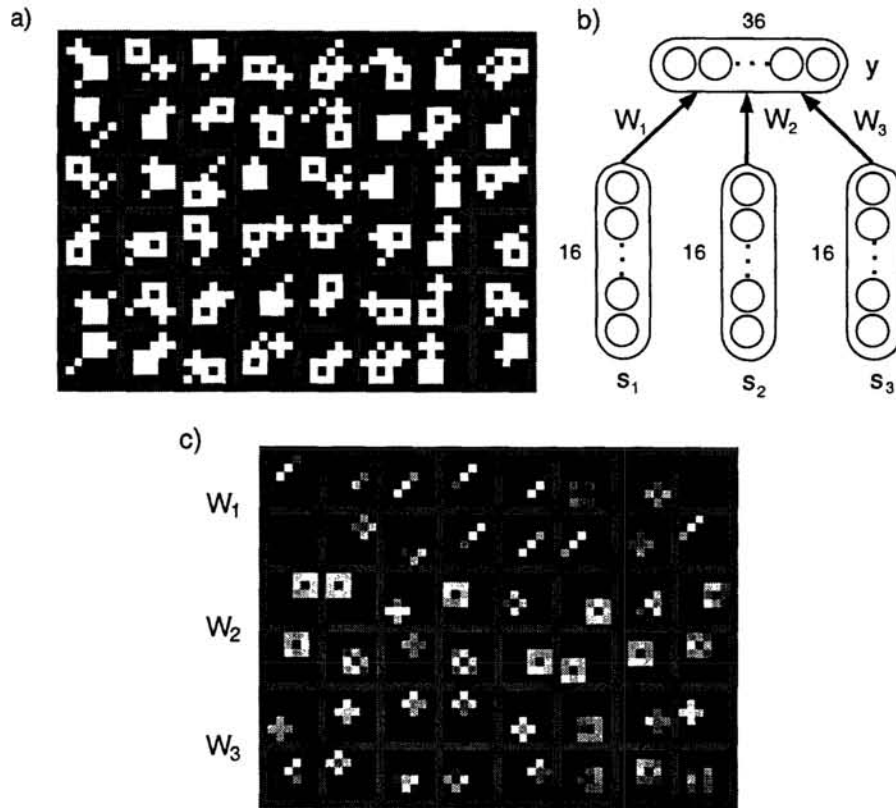

Figure 3. Shapes Problem. a) Sample images from the data set. b) Learning architecture used. c) Output weight matrix after learning.

mean-field equations also performs quite well.

## 5   Discussion

The factorial learning problem for cooperative vector quantizers has been formulated in the EM framework, and two learning algorithms, based on Gibbs sampling and mean-field approximation, have been derived. Unlike the Boltzmann machine, Gibbs sampling for this architecture seems to require very few samples for adequate performance. This may be due to the fact that, whereas the Boltzmann machine relies on *differences* of noisy estimates for its weight changes, due to the constant partition function the factorial learning algorithm does not. The mean-field approximation also seems to perform quite well on all problems tested to date. This may also be a consequence of the constant partition function which guarantees that the mean-field cost is an upper bound on the exact cost.

The framework can be extended to hidden Markov models (HMMs), showing that simple HMMs are a special case of dynamical CVQs, with the general case corresponding to parallel, factorial HMMs. The two principal advantages of such architectures are (1) unlike the traditional HMM, the state space can be represented as a combination of features, and (2) time series generated by multiple sources can be modeled. Simulation results on the Gibbs and mean-field EM algorithms for factorial HMMs are also promising (Ghahramani, 1995).

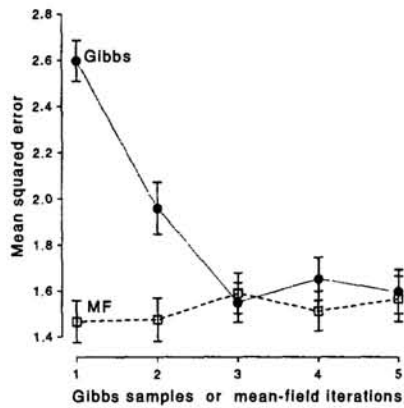

Figure 4. Comparison of the Gibbs and mean-field EM algorithms for the lines data. Each data point shows the mean squared training error averaged over 10 runs of 20 EM steps, with standard error bars. For the Gibbs curve the abscissa is the number of samples per vector of hidden units; for the mean-field curve it is the number of iterations of equation (5).

## Acknowledgements

The author wishes to thank Lawrence Saul and Michael Jordan for invaluable discussions. This project was supported in part by a grant from the McDonnell-Pew Foundation, by a grant from ATR Human Information Processing Research Laboratories, by a grant from Siemens Corporation, and by grant N00014-94-1-0777 from the Office of Naval Research.

## Footnotes

[1] For the remainder of the paper we will ignore the second term in (1), thereby assuming equal priors on the hidden states. Relaxing this assumption and estimating priors from the data is straightforward.

## References

Ackley, D., Hinton, G., and Sejnowski, T. (1985). A learning algorithm for Boltzmann machines. *Cognitive Science*, 9:147–169.

Barlow, H. (1989). Unsupervised learning. *Neural Computation*, 1:295–311.

Baum, L., Petrie, T., Soules, G., and Weiss, N. (1970). A maximization technique occurring in the statistical analysis of probabilistic functions of Markov chains. *The Annals of Mathematical Statistics*, 41:164–171.

Dempster, A., Laird, N., and Rubin, D. (1977). Maximum likelihood from incomplete data via the EM algorithm. *J. Royal Statistical Society Series B*, 39:1–38.

Geman, S. and Geman, D. (1984). Stochastic relaxation, Gibbs distributions, and the Bayesian restoration of images. *IEEE Transactions on Pattern Analysis and Machine Intelligence*, 6:721–741.

Ghahramani, Z. (1995). Factorial learning and the *EM* algorithm. *MIT Computational Cognitive Science TR 9501*.

Hinton, G. and Zemel, R. (1994). Autoencoders, minimum description length, and Helmholtz free energy. In Cowan, J., Tesauro, G., and Alspector, J., editors, *Advances in Neural Information Processing Systems 6*. Morgan Kaufmanm Publishers, San Francisco, CA.

Jordan, M. and Jacobs, R. (1994). Hierarchical mixtures of experts and the EM algorithm. *Neural Computation*, 6:181–214.

McCullagh, P. and Nelder, J. (1989). *Generalized Linear Models*. Chapman & Hall, London.

Neal, R. (1992). Connectionist learning of belief networks. *Artificial Intelligence*, 56:71–113.

Parisi, G. (1988). *Statistical Field Theory*. Addison-Wesley, Redwood City, CA.

Redlich, A. (1993). Supervised factorial learning. *Neural Computation*, 5:750–766.

Rissanen, J. (1989). *Stochastic Complexity in Statistical Inquiry*. World Scientific, Singapore.

Saund, E. (1995). A multiple cause mixture model for unsupervised learning. *Neural Computation*, 7(1):51–71.

Zemel, R. (1993). *A minimum description length framework for unsupervised learning*. Ph.D. Thesis, Dept. of Computer Science, University of Toronto, Toronto, Canada.
